# Constructing Heterogeneous Committees Using Input Feature Grouping: Application to Economic Forecasting

**Yuansong Liao and John Moody**
Department of Computer Science, Oregon Graduate Institute,
P.O.Box 91000, Portland, OR 97291-1000

## Abstract

The committee approach has been proposed for reducing model uncertainty and improving generalization performance. The advantage of committees depends on (1) the performance of individual members and (2) the correlational structure of errors between members. This paper presents an input grouping technique for *designing* a *heterogeneous committee*. With this technique, all input variables are first grouped based on their mutual information. Statistically similar variables are assigned to the same group. Each member's input set is then formed by input variables extracted from different groups. Our *designed committees* have less error correlation between its members, since each member observes different input variable combinations. The individual member's feature sets contain less redundant information, because highly correlated variables will not be combined together. The member feature sets contain almost complete information, since each set contains a feature from each information group. An empirical study for a noisy and nonstationary economic forecasting problem shows that committees constructed by our proposed technique outperform committees formed using several existing techniques.

## 1 Introduction

The committee approach has been widely used to reduce model uncertainty and improve generalization performance. Developing methods for generating candidate committee members is a very important direction of committee research. Good candidate members of a committee should have (1) good (not necessarily excellent) individual performance and (2) small residual error correlations with other members.

Many techniques have been proposed to reduce residual correlations between members. These include resampling the training and validation data [3], adding randomness to data [7], and decorrelation training [8]. These approaches are only effective for certain models and problems. Genetic algorithms have also been used to generate good and diverse members [6].

Input feature selection is one of the most important stages of the model learning process. It has a crucial impact both on the learning complexity and the general-

ization performance. It is essential that a feature vector gives sufficient information for estimation. However, too many redundant input features not only burden the whole learning process, but also degrade the achievable generalization performance.

Input feature selection for individual estimators has received a lot of attention because of its importance. However, there has not been much research on feature selection for estimators in the context of committees. Previous research found that giving committee members different input features is very useful for improving committee performance [4], but is difficult to implement [9]. The feature selection problem for committee members is conceptually different than for single estimators. When using committees for estimation, as we stated previously, committee members not only need to have reasonable performance themselves, but should also make decisions independently.

When all committee members are trained to model the same underlying function, it is difficult for committee members to optimize both criteria at the same time. In order to generate members that provide a good balance between the two criteria, we propose a feature selection approach, called **input feature grouping**, for committee members. The idea is to give each member estimator of a committee a rich but distinct feature sets, in the hope that each member will generalize independently with reduced error correlations.

The proposed method first groups input features using a hierarchical clustering algorithm based on their mutual information, such that features in different groups are less related to each other and features within a group are statistically similar to each other. Then the feature set for each committee member is formed by selecting a feature from each group. Our empirical results demonstrate that forming a heterogeneous committee using input feature grouping is a promising approach.

## 2   Committee Performance Analysis

There are many ways to construct a committee. In this paper, we are mainly interested in heterogeneous committees whose members have different input feature sets. Committee members are given different subsets of the available feature set. They are trained independently, and the committee output is either a weighted or unweighted combination of individual members' outputs.

In the following, we analyze the relationship between committee errors and average member errors from the regression point of view and discuss how the residual correlations between members affect the committee error. We define the training data $\mathcal{D} = \{(X^\beta, Y^\beta); \beta = 1, 2, \ldots N\}$ and the test data $\mathcal{T} = \{(X^\mu, Y^\mu); \mu = 1, 2, \ldots \infty\}$, where both are assumed to be generated by the model: $Y = t(X) + \epsilon$, $\epsilon \sim \mathcal{N}(0, \sigma^2)$. The data $\mathcal{D}$ and $\mathcal{T}$ are independent, and inputs are drawn from an unknown distribution. Assume that a committee has $K$ members. Denote the available input features as $X = [x_1, x_2, \ldots, x_m]$, the feature sets for the $i^{th}$ and $j^{th}$ members as $X_i = [x_{i_1}, x_{i_2}, \ldots, x_{m_i}]$ and $X_j = [x_{j_1}, x_{j_2}, \ldots, x_{m_j}]$ respectively, where $X_i \in X$, $X_j \in X$ and $X_i \neq X_j$, and the mapping function of the $i^{th}$ and $j^{th}$ member models trained on data from $\mathcal{D}$ as $f_i(X_i)$ and $f_j(X_j)$. Define the *model error* $e_i^\mu = t^\mu - f_i(X_i^\mu)$, for all $\mu = 1, 2, 3, \ldots, \infty$ and $i = 1, 2, \ldots, K$.

The MSE of a committee is

$$E_C = \mathcal{E}_\mu\left[\left(t^\mu - \frac{1}{K}\sum_{i=1}^{K} f_i(X_i^\mu)\right)^2\right] = \frac{1}{K^2}\sum_{i=1}^{K}\mathcal{E}_\mu[(e_i^\mu)^2] + \frac{1}{K^2}\sum_{i\neq j}^{K}\mathcal{E}_\mu[e_i^\mu e_j^\mu] \ , \quad (1)$$

and the average MSE made by the committee members acting individually is

$$E_{\text{ave}} = \frac{1}{K}\sum_{i=1}^{K}\mathcal{E}_\mu[(e_i^\mu)^2] \ , \quad (2)$$

where $\mathcal{E}[\cdot]$ denotes the expectation over all test data $\mathcal{T}$. Using Jensen's inequality, we get $E_C \leq E_{\text{ave}}$, which indicates that the performance of a committee is always equal to or better than the average performance of its members.

We define the *average model error correlation* as $C = \frac{1}{K(K-1)}\sum_{i\neq j}^{K}\mathcal{E}_\mu[e_i^\mu e_j^\mu]$ , and then have

$$E_C = \frac{1}{K}E_{\text{ave}} + \frac{K-1}{K}C = (\frac{1}{K} + \frac{K-1}{K}q)E_{\text{ave}} \ , \quad (3)$$

where $q = \frac{C}{E_{\text{ave}}}$ . We consider the following four cases of $q$:

- **Case 1:** $-\frac{1}{K-1} \leq q < 0$. In this case, the model errors between members are anti-correlated, which might be achieved through decorrelation training.

- **Case 2:** $q = 0$. In this case, the *model errors* between members are uncorrelated, and we have: $E_C = \frac{1}{K}E_{\text{ave}}$. That is to say, a committee can do much better than the average performance of its members.

- **Case 3:** $0 < q < 1$. If $E_{\text{ave}}$ is bounded above, when the committee size $K \longrightarrow \infty$, we have $E_C = qE_{\text{ave}}$ . This gives the asymptotic limit of a committee's performance. As the size of a committee goes to infinity, the committee error is equal to the average model error correlation $C$. The difference between $E_C$ and $E_{\text{ave}}$ is determined by the ratio $q$.

- **Case 4:** $q = 1$. In this case, $E_C$ is equal to $E_{\text{ave}}$. This happens only when $e_i = e_j$, for all $i,j = 1,\dots,K$. It is obvious that there is no advantage to combining a set of models that act identically.

It is clear from the analyses above that a committee shows its advantage when the ratio $q$ is less than one. The smaller the ratio $q$ is, the better the committee performs compared to the average performance of its members. For the committee to achieve substantial improvement over a single model, committee members not only should have small errors individually, but also should have small residual correlations between each other.

## 3 Input Feature Grouping

One way to construct a feature subset for a committee member is by randomly picking a certain number of features from the original feature set. The advantage of this method is that it is simple. However, we have no control on each member's performance or on the residual correlation between members by randomly selecting subsets.

Instead of randomly picking a subset of features for a member, we propose an input feature grouping method for forming committee member feature sets. The input grouping method first groups features based on a relevance measure in a way such that features between different groups are less related to one another and features within a group are more related to one another.

After grouping, there are two ways to form member feature sets. One method is to construct the feature set for each member by selecting a feature from each group. Forming a member's feature set in this way, each member will have enough information to make decision, and its feature set has less redundancy. This is the method we use in this paper.

Another way is to use each group as the feature set for a committee member. In this method each member will only have partial information. This is likely to hurt individual member's performance. However, because the input features for different members are less dependent, these members tend to make decisions more independently. There is always a trade-off between increasing members' independence and hurting individual members' performance. If there is no redundancy among input feature representations, removing several features may hurt individual members' performance badly, and the overall committee performance will be hurt even though members make decisions independently. This method is currently under investigation.

The mutual information $I(x_i; x_j)$ between two input variables $x_i$ and $x_j$ is used as the relevance measure to group inputs. The mutual information $I(x_i; x_j)$, which is defined in equation 4, measures the dependence between the two random variables.

$$I(x_i; x_j) = H(x_i) - H(x_i|x_j) = \sum_{x_i, x_j} p(x_i, y_i) \log \frac{p(x_i, x_j)}{p(x_i)p(x_j)} \ . \qquad (4)$$

If features $x_i$ and $x_j$ are highly dependent, $I(x_i; x_j)$ will be large. Because the mutual information measures arbitrary dependencies between random variables, it has been effectively used for feature selections in complex prediction tasks [1], where methods bases on linear relations like the correlation are likely to make mistakes. The fact that the mutual information is independent of the coordinates chosen permits a robust estimation.

## 4 Empirical Studies

We apply the input grouping method to predict the one-month rate of change of the Index of Industrial Production (IP), one of the key measures of economic activity. It is computed and published monthly. Figure 4 plots monthly IP data from 1967 to 1993.

Nine macroeconomic time series, whose names are given in Table 1, are used for forecasting IP. Macroeconomic forecasting is a difficult task because data are usually limited, and these series are intrinsically very noise and nonstationary. These series are preprocessed before they are applied to the forecasting models. The representation used for input series is the first difference on one month time scales of the logged series. For example, the notation IP.L.D1 represents IP.L.D1 $\equiv ln(\text{IP}(t)) - ln(\text{IP}(t\text{-}1))$. The target series is IP.L.FD1, which is defined as IP.L.FD1 $\equiv ln(\text{IP}(t\text{+}1)) - ln(\text{IP}(t))$. The data set has been one of our benchmarks for various studies [5, 10].

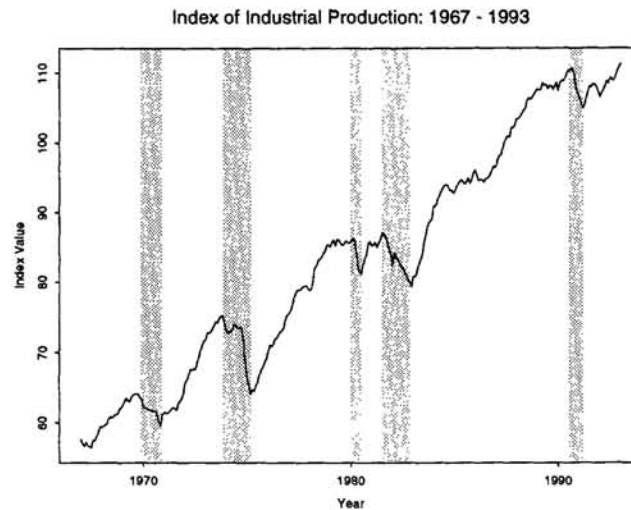

Figure 1: U.S. Index of Industrial Production (IP) for the period 1967 to 1993. Shaded regions denote official recessions, while unshaded regions denote official expansions. The boundaries for recessions and expansions are determined by the National Bureau of Economic Research based on several macroeconomic series. As is evident for IP, business cycles are irregular in magnitude, duration, and structure, making prediction of IP an interesting challenge.

| Series | Description |
|--------|-------------|
| IP | Index of Industrial Production |
| SP | Standard & Poor's 500 |
| DL | Index of Leading Indicators |
| M2 | Money Supply |
| CP | Consumer Price Index |
| CB | Moody's Aaa Bond Yield |
| HS | Housing Starts |
| TB3 | 3-month Treasury Bill Yield |
| Tr | Yield Curve Slope: (10-Year Bond Composite)-(3-Month Treasury Bill) |

Table 1: Input data series. Data are taken from the Citibase database.

During the grouping procedure, measures of mutual information between all pairs of input variables are computed first. A simple histogram method is used to calculate these estimates. Then a hierarchical clustering algorithm [2] is applied to these values to group inputs. Hierarchical clustering proceeds by a series of successive fusions of the nine input variables into groups. At any particular stage, the process fuses variables or groups of variables which are closest, base on their mutual information estimates. The distance between two groups is defined as the average of the distances between all pairs of individuals in the two groups. The result is presented by a tree which illustrates the fusions made at each successive level (see Figure 2). From the clustering tree, it is clear that we can break the input variables into four groups: (IP.L.D1, DL.L.D1) measure recent economic changes, (SP.L.D1) reflects recent stock market momentum, (CB.D1, TB3.D1, Tr.D1) give interest rate information, and (M2.L.D1, CP.L.D1, HS.L.D1) provide inflation information. The grouping algorithm meaningfully clusters the nine input series.

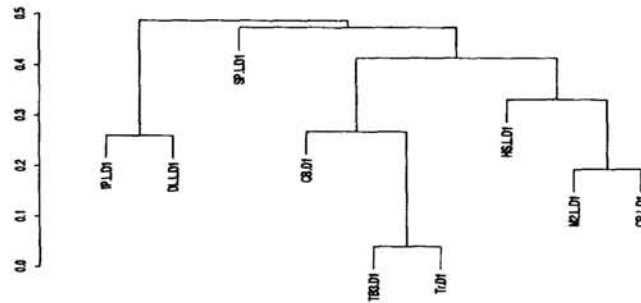

Figure 2: Variable grouping based on mutual information. Y label is the distance.

Eighteen different subsets of features can be generated from the four groups by selecting a feature from each group. Each subset is given to a committee member. For example, the subsets (IP.L.D1, SP.L.D1, CB.D1, M2.L.D1) and (DL.L.D1, SP.L.D1, TB3.D1, M2.L.D1) are used as feature sets for different committee members. A committee has totally eighteen members. Data from Jan. 1950 to Dec. 1979 is used for training and validation, and from Jan. 1980 to Dec. 1989 is used for testing. Each member is a linear model that is trained using neural net techniques.

We compare the input grouping method with three other committee member generating methods: baseline, random selection, and bootstrapping. The baseline method is to train a committee member using all the input variables. Members are only different in their initial weights. The bootstrapping method also trains a member using all the input features, but each member has different bootstrap replicates of the original training data as its training and validation sets. The random selection method constructs a feature set for a member by randomly picking a subset from the available features. For comparison with the grouping method, each committee generated by these three methods has 18 members.

Twenty runs are performed for each of the four methods in order to get reliable performance measures. Figure 3 shows the boxplots of normalized MSE for the four methods. The grouping method gives the best result, and the performance improvement is significant compared to other methods. The grouping method outperforms the random selection method by meaningfully grouping of input features. It is interesting to note that the heterogeneous committee methods, grouping and random selection, perform better than homogeneous methods for this data set. One of the reasons for this is that giving different members different input sets increases their model independence. Another reason could be that the problem becomes easier to model because of smaller feature sets.

## 5  Conclusions

The performance of a committee depends on both the performance of individual members and the correlational structure of errors between members. An empirical study for a noisy and nonstationary economic forecasting problem has demonstrated that committees constructed by input variable grouping outperform committees formed by randomly selecting member input variables. They also outperform committees without any input variable manipulation.

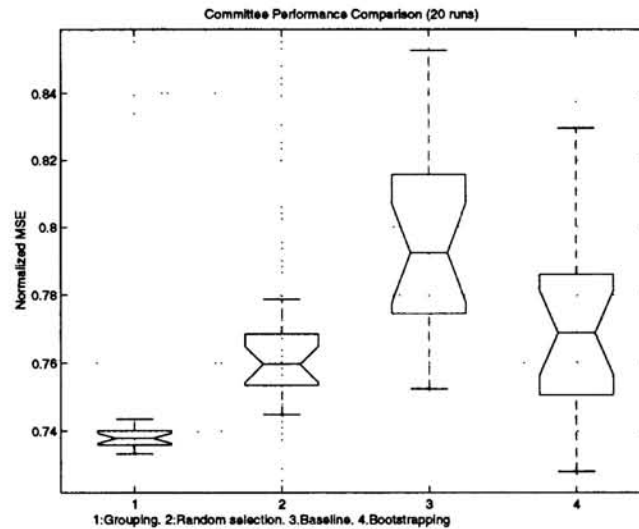

Figure 3: Comparison between four different committee member generating methods. The proposed grouping method gives the best result, and the performance improvement is significant compared to the other three methods.

# References

[1] R. Battiti. Using mutual information for selecting features in supervised neural net learning. *IEEE Trans. on Neural Networks*, 5(4), July 1994.

[2] B.Everitt. *Cluster Analysis*. Heinemann Educational Books, 1974.

[3] L. Breiman. Bagging predictors. *Machine Learning*, 24(2):123–40, 1996.

[4] K.J. Cherkauer. Human expert-level performance on a scientific image analysis task by a system using combined artifical neural networks. In P. Chan, editor, *Working Notes of the AAAI Workshop on Integrating Multiple Learned Models*, pages 15–21. 1996.

[5] J. Moody, U. Levin, and S. Rehfuss. Predicting the U.S. index of industrial production. *In proceedings of the 1993 Parallel Applications in Statistics and Economics Conference, Zeist, The Netherlands. Special issue of* Neural Network World, 3(6):791–794, 1993.

[6] D. Opitz and J. Shavlik. Generating accurate and diverse members of a neural-network ensemble. In D. Touretzky, M. Mozer, and M. Hasselmo, editors, *Advances in Neural Information Processing Systems 8*. MIT Press, Cambridge, MA, 1996.

[7] Y. Raviv and N. Intrator. Bootstrapping with noise: An effective regularization technique. *Connection Science*, 8(3-4):355–72, 1996.

[8] B. E. Rosen. Ensemble learning using decorrelated neural networks. *Connection Science*, 8(3-4):373–83, 1996.

[9] K. Tumer and J. Ghosh. Error correlation and error reduction in ensemble classifiers. *Connection Science*, 8(3-4):385–404, December 1996.

[10] L. Wu and J. Moody. A smoothing regularizer for feedforward and recurrent neural networks. *Neural Computation*, 8.3:463–491, 1996.
